# Generalization Properties of Radial Basis Functions

Sherif M. Botros          Christopher G. Atkeson
Brain and Cognitive Sciences Department
and the Artificial Intelligence Laboratory
Massachusetts Institute of Technology
Cambridge, MA 02139

## Abstract

We examine the ability of radial basis functions (RBFs) to generalize. We compare the performance of several types of RBFs. We use the inverse dynamics of an idealized two-joint arm as a test case. We find that without a proper choice of a norm for the inputs, RBFs have poor generalization properties. A simple global scaling of the input variables greatly improves performance. We suggest some efficient methods to approximate this distance metric.

## 1  INTRODUCTION

The Radial Basis Functions (RBF) approach to approximating functions consists of modeling an input-output mapping as a linear combination of radially symmetric functions (Powell, 1987; Poggio and Girosi, 1990; Broomhead and Lowe, 1988; Moody and Darken, 1989). The RBF approach has some properties which make it attractive as a function interpolation and approximation tool. The coefficients that multiply the different basis functions can be found with a linear regression. Many RBFs are derived from regularization principles which optimize a criterion combining fitting error and the smoothness of the approximated function. However, the optimality criteria may not always be appropriate, especially when the input variables have different measurement units and affect the output differently. A natural extension to RBFs is to vary the distance metric (equivalent to performing a linear transformation on the input variables). This can be viewed as changing the cost function to be optimized (Poggio and Girosi, 1990). We first use an exact interpolation approach with RBFs centered at the data in the training set. We then explore the effect of optimizing the distance metric for Gaussian RBFs using a

smaller number of functions than data in the training set. We also suggest and test different methods to approximate this metric for the case of Gaussian RBFs that work well for the two joint arm example that we examined. We refer the reader to several other studies addressing the generalization performance of RBFs (Franke, 1982; Casdagli, 1989; Renals and Rohwer, 1989).

## 2   EXACT INTERPOLATION APPROACH

In the exact interpolation model the number of RBFs is equal to the number of experiences. The centers of the RBFs are chosen to be at the location of the experiences. We used an idealized horizontal planar two joint arm model with no friction and no noise (perfect measurements) to test the performance of RBFs:

$$
\begin{aligned}
T_1 &= \ddot{\theta}_1(I_1 + I_2 + 2m_2 c_{x_2} l_1 \cos\theta_2 - 2m_2 c_{y_2} l_1 \sin\theta_2) \\
&\quad + \ddot{\theta}_2(I_2 + m_2 c_{x_2} l_1 \cos\theta_2 - m_2 c_{y_2} l_1 \sin\theta_2) \\
&\quad - 2l_1 \dot{\theta}_1 \dot{\theta}_2 (m_2 c_{x_2} \sin\theta_2 + m_2 c_{y_2} \cos\theta_2) \\
&\quad - l_1 \dot{\theta}_2^{\,2} (m_2 c_{x_2} \sin\theta_2 + m_2 c_{y_2} \cos\theta_2) \\
T_2 &= \ddot{\theta}_1(m_2 c_{x_2} l_1 \cos\theta_2 - m_2 c_{y_2} l_1 \sin\theta_2 + I_2) + \ddot{\theta}_2 I_2 \\
&\quad + l_1 \dot{\theta}_1^{\,2} (m_2 c_{x_2} \sin\theta_2 + m_2 c_{y_2} \cos\theta_2)
\end{aligned}
\tag{1}
$$

where $\theta_i$, $\dot{\theta}_i$, $\ddot{\theta}_i$ are the angular position, velocity and acceleration of joint $i$. $T_i$ is the torque at joint $i$. $I_i$, $m_i$, $l_i$, $c_{x_i}$ and $c_{y_i}$ are respectively the moment of inertia, mass, length and the x and y components of the center of mass location of link $i$. The input vector is $(\theta_1, \theta_2, \dot{\theta}_1, \dot{\theta}_2, \ddot{\theta}_1, \ddot{\theta}_2)$. The training and test sets are formed of one thousand random experiences each; uniformly distributed across the space of the inputs. The different inputs were selected from the following ranges: [-4, 4] for the joint angles, [-20, 20] for the joint angular velocities and [-100, 100] for the joint angular accelerations. For the exact interpolation case, we scaled the input variables such that the input space is limited to the six dimensional hypercube $[-1, 1]^6$. This improved the results we obtained. The torque to be estimated at each joint is modeled by the following equation:

$$
T_k(\mathbf{x}_i) = \sum_{j=1}^{n} C_{kj} \phi(\|\mathbf{x}_i - \mathbf{x}_j\|) + \sum_{j=1}^{p} \mu_{kj} P_j^m(\mathbf{x}_i)
\tag{2}
$$

where $T_k, k = 1, 2$, is the estimated torque at the $k^{th}$ joint, $n$ is the number of experiences/RBFs, $\mathbf{x}_i$ is the $i^{th}$ input vector, $\| \cdot \|$ is the Euclidean norm and $P_j^m(\cdot), j = 1, \ldots, p$, is the space of polynomials of order $m$. The polynomial terms are not always added and it was found that adding the polynomial terms by themselves does not improve the performance significantly, which is in agreement with the conclusion made by Franke (Franke, 1982). When a polynomial is present in the equation, we add the following extra constraints (Powell, 1987):

$$
\sum_{j=1}^{n} C_{kj} P_i^m(\mathbf{x}_j) = 0 \qquad i = 1, \ldots, p
\tag{3}
$$

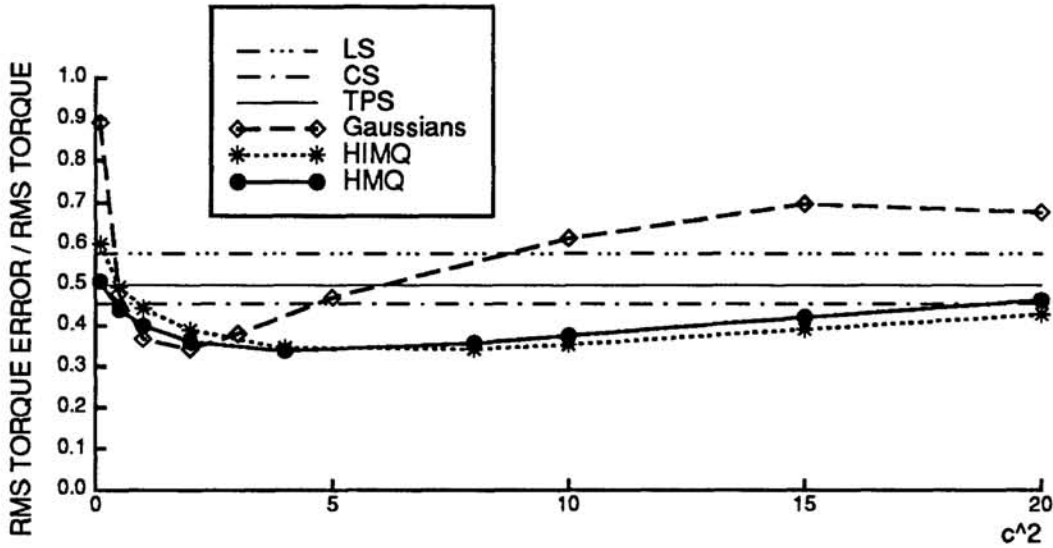

Figure 1: Normalized errors for the different RBFs using exact interpolation. $c^2$ is the width parameter when relevant.

To find the coefficients $C_{kj}$ and $\mu_{kj}$ , we have to invert a square matrix which is nonsingular for distinct inputs for the basis functions we considered (Micchelli, 1986). We used the training set to find the parameters $C_{kj}$, $j = 1, n,$ and when relevant $\mu_{kj}$, $j = 1, p,$ for the following RBFs:

- Gaussians   $\phi(r) = \exp(\frac{-r^2}{c^2})$
- Hardy Multiquadrics [HMQ]   $\phi(r) = \sqrt{r^2 + c^2}$
- Hardy Inverse Multiquadrics [HIMQ]   $\phi(r) = \frac{1}{\sqrt{r^2 + c^2}}$
- Thin Plate Splines [TPS]   $\phi(r) = r^2 \log r$
- Cubic Splines [CS]   $\phi(r) = r^3$
- Linear Splines [LS]   $\phi(r) = r$

where $r = \|\mathbf{x}_i - \mathbf{x}_j\|$ . For the last three RBFs, we added polynomials of different orders, subject to the constraints in equation 3 above. Since the number of independent parameters is equal to the number of points in the training set, we can train the system so that it exactly reproduces the training set. We then tested its performance on the test set. The error was measured by equation 4 below:

$$E = \sqrt{\frac{\sum_{k=1}^{2} \sum_{i=1}^{n} (T_{ki} - \hat{T}_{ki})^2}{\sum_{k=1}^{2} \sum_{i=1}^{n} T_{ki}^2}} \qquad (4)$$

The normalized error obtained on the test set for the different RBFs are shown in figure 1. The results for LS and CS shown in this figure are obtained after the addition of a first order polynomial to the RBFs. We also tried adding a third order polynomial for TPS. As shown in this figure, the normalized error was more sensitive to the width parameter (i.e. $c^2$) for the Gaussian RBFs than for

Hardy multiquadrics and inverse multiquadrics. This is in agreement with Franke's observation (Franke, 1982). The best normalized error for any RBF that we tested was 0.338 for HMQ with a value of $c^2 = 4$ . Also, contrary to our expectations and to results reported by others (Franke, 1982), the TPS with a third order polynomial had a normalized error of 0.5003. This error value did not change significantly when only lower order polynomials are added to the $(r^2 \log r)$ RBFs. Using Generalized Cross Validation (Bates *et al.*, 1987) to optimize the tradeoff between smoothness and fitting the data, we got similar normalized error for TPS.

## 3   GENERALIZED RBF

The RBF approach has been generalized (Poggio and Girosi, 1990) to have adjustable center locations, fewer centers than data, and to use a different distance metric. Instead of using a Euclidean norm, we can use a general weighted norm:

$$\|\mathbf{x}_i - \mathbf{x}_j\|_W^2 = (\mathbf{x}_i - \mathbf{x}_j)^T \mathbf{W}^T \mathbf{W}(\mathbf{x}_i - \mathbf{x}_j) \tag{5}$$

where $\mathbf{W}$ is a square matrix. This approach is also referred to as Hyper Basis Functions (Poggio and Girosi, 1990). The problem of finding the weight matrix and the location of the centers is nonlinear. We simplified the problem by only considering a diagonal matrix $\mathbf{W}$ and fixing the locations of the centers of the RBFs. The center locations were chosen randomly and were uniformly distributed over the input space. We tested three different methods to find the different parameters for Gaussian RBFs that we will describe in the next three subsections.

### 3.1   NONLINEAR OPTIMIZATION

We used a Levenberg-Marquardt nonlinear optimization routine to find the coefficients of the RBFs $\{C_i\}$ and the diagonal scaling matrix $\mathbf{W}$ that minimized the sum of the squares of the errors in estimating the training set. We were able to find a set of parameters that reduced the normalized error to less than 0.01 in both the training and the test sets using 500 Gaussian RBFs randomly and uniformly spaced over the input space. One disadvantage we found with this method is the possible convergence to local minima and the long time it takes to converge using general purpose optimization programs. The diagonal elements of the matrix $\mathbf{W}$ are shown in the L-M columns of table 1. As expected, $\theta_1$ has a very small scale for both joints compared to $\theta_2$, since $\theta_1$ does not affect the output of either joint in the horizontal model described by equation 1. Also the scaling of $\theta_2$ is much larger than the scaling of the other variables. This suggests that the scaling could be dependent on both the range of the input variables as well as the sensitivity of the output to the different input variables. We found empirically that a formula of the form of equation 6 approximates reasonably well the scaling weights found using nonlinear optimization.

$$w_{ii} = \frac{\overline{|\nabla f_i|}}{\|\nabla f\|} \frac{k}{\sqrt{E\{(x_i - t_i)^2\}}} \tag{6}$$

where $\frac{\overline{|\nabla f_i|}}{\|\nabla f\|}$ is the normalized average absolute value of the gradient of the correct model of the function to be approximated. The term $\frac{k}{\sqrt{E\{(x_i - t_i)^2\}}}$ normalizes the

Table 1: Scaling Weights Obtained Using Different Methods.

| W | L-M ALG. | | TRUE FUNC. | | GRAD. APPROX. | |
|---|---|---|---|---|---|---|
| | Joint 1 | Joint 2 | Joint 1 | Joint 2 | Joint 1 | Joint 2 |
| $W_{11}(\theta_1)$ | 0.000021 | 5.48237e-06 | 0.000000 | 0.000000 | 0.047010 | 0.005450 |
| $W_{22}(\theta_2)$ | 0.382014 | 0.443273 | 0.456861 | 0.456449 | 0.400615 | 0.409277 |
| $W_{33}(\dot{\theta}_1)$ | 0.004177 | 0.0871921 | 0.005531 | 0.010150 | 0.009898 | 0.038288 |
| $W_{44}(\dot{\theta}_2)$ | 0.004611 | 0.000120948 | 0.007490 | 0.000000 | 0.028477 | 0.008948 |
| $W_{55}(\ddot{\theta}_1)$ | 0.000433 | 0.00134168 | 0.000271 | 0.000110 | 0.006365 | 0.002166 |
| $W_{66}(\ddot{\theta}_2)$ | 0.000284 | 0.000955884 | 0.000059 | 0.000116 | 0.000556 | 0.001705 |

density of the input variables in each direction by taking into account the expected distances from the RBF centers to the data. The constant $k$ in this equation is inversely proportional to the width of the Gaussian used in the RBF. For the inverse dynamics problem we tested, and using 500 Gaussian functions randomly and uniformly distributed over the entire input space, a $k$ between 1 and 2 was found to be good and results in scaling parameters which approximate those obtained by optimization. The scaling weights obtained using equation 6 and based on the knowledge of the functions to be approximated are shown in the TRUE FUNC. columns of table 1. Using these weight values the error in the test set was about 0.0001

## 3.2   AVERAGE GRADIENT APPROXIMATION

In the previous section we showed that the scaling weights could be approximated using the derivatives of the function to be approximated in the different directions. If we can approximate these derivatives, we can then approximate the scaling weights using equation 6. A change in the output $\Delta y$ could be approximated by a first order Taylor series expansion as shown in equation 7 below:

$$\Delta y = \sum_{i=1}^{d} \frac{\partial y}{\partial x_i} \Delta x_i \tag{7}$$

We first scaled the input variables so that they have the same range, then selected all pairs of points from the training set that are below a prespecified distance (since equation 7 is only valid for nearby points), and then computed $\Delta x_i$ and $\Delta y$ for each pair. We used least squares regression to estimate the values of $\frac{\partial y}{\partial x_i}$. Using the estimated derivatives and equation 6, we got the scaling weights shown in the last two columns of table 1. Note the similarity between these weights and the ones obtained using the nonlinear optimization or the derivatives of the true function. The normalized error in this case was found to be 0.012 for the training set and 0.033 for the test set. One advantage of this method is that it is much faster than the nonlinear optimization method. However, it is less accurate.

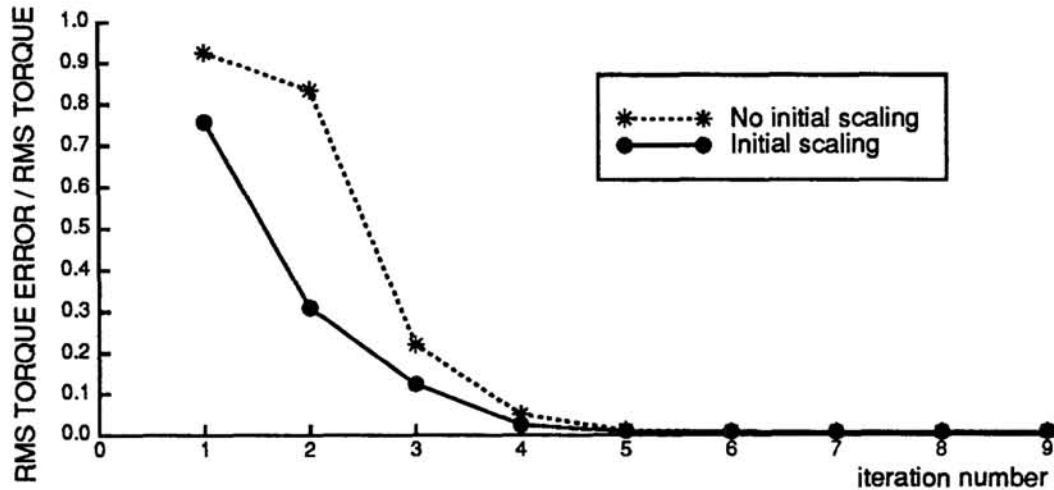

Figure 2: Normalized error vs. the number of iterations using the recursive method with and without initial scaling.

## 3.3    RECURSIVE METHOD

Another possible method to approximate the RMS values of the derivatives is to first approximate the function using RBFs of "reasonable" widths and then use this first approximation to compute the derivatives which are then used to modify the Gaussian widths in the different directions. The procedure is then repeated. We used 100 RBFs with Gaussian units randomly and uniformly distributed to find the coefficients of the RBFs. We explored two different scalings of the input data. In the first case we used the raw data without scaling, and in the second case the different input variables were scaled so that they have the same range from [-1, 1]. The width of the Gaussians used as specified by the variance $c^2$ was equal to 200 in the first case, and 2 in the second case. We then used the estimated values of the derivatives to change the width of the Gaussians in the different directions and iterated the procedure. The normalized error is plotted versus the number of iterations for both cases in figure 2. As shown in this figure, the test set error dropped to around 0.001 in about only 4 iterations. This technique is also much faster than the nonlinear optimization approach. Also it can be easily made local, which is desirable if the dependence of the function to be approximated on the input variables changes from one region of the input space to the other. One disadvantage of this approach is that it is not guaranteed to converge especially if the initial approximation of the function is very bad.

## 4    CONCLUSION

In this paper we tested the ability of RBFs to generalize from the training set. We found that the choice of the distance metric used may be crucial for good generalization. For the problem we tested, a bad choice of a distance metric resulted in very poor generalization. However, the performance of Gaussian RBFs improved significantly if we optimized the distance metric. We also tested some empirical

methods for efficiently estimating this metric that worked well in our test problem. Additional work has to be done to identify the conditions under which the techniques we presented here may or may not work. Although a simple global scaling of the input variables worked well in our test example, it may not work in general. One problem that we found when we optimized the distance metric is that the values of the coefficients $C_i$ become very large, even if we imposed a penalty on their values. The reason for this, we think, is that the estimation problem was close to singular. Choice of the training set and optimizing the centers of the RBFs may solve this problem. The recursive method we described could probably be modified to approximate a complete linear coordinate transformation and local scaling.

## Acknowledgments

Support was provided under Office of Naval Research contract N00014-88-K-0321 and under Air Force Office of Scientific Research grant AFOSR-89-0500. Support for CGA was provided by a National Science Foundation Engineering Initiation Award and Presidential Young Investigator Award, an Alfred P. Sloan Research Fellowship, the W. M. Keck Foundation Assistant Professorship in Biomedical Engineering, and a Whitaker Health Sciences Fund MIT Faculty Research Grant.

## References

D. M. Bates, M. J. Lindstorm, G. Wahba and B. S. Yandel (1987) "GCVPACK - Routines for generalized cross validation". *Commun. Statist.-Simulat.* **16** (1): 263-297.

D. S. Broomhead and D. Lowe (1988) "Multivariable functional interpolation and adaptive networks". *Complex Systems* **2**:321-323.

M. Casdagli (1989) "Nonlinear prediction of chaotic time series". *Physica D* **35**: 335-356.

R. Franke (1982) "Scattered data interpolation: Tests of some methods". *Math. Comp.* **38**(5):181-200.

C. A. Micchelli (1986) "Interpolation of scattered data: distance matrices and conditionally positive definite functions". *Constr. Approx.* **2**:11-22.

J. Moody and C. Darken (1989) "Fast learning in networks of locally tuned processing units". *Neural Computation* **1**(2):281-294.

T. Poggio and F. Girosi (1990) "Networks for approximation and learning". *Proceedings of the IEEE* **78**(9):1481-1497.

M. J. D. Powell (1987) "Radial basis functions for multivariable interpolation: A review". In J. C. Mason and M. G. Cox (ed.), *Algorithms for Approximation*, 143-167. Clarendon Press, Oxford.

S. Renals and R. Rohwer (1989) "Phoneme classification experiments using radial basis functions". In *Proceedings of the International Joint Conference on Neural Networks*, I-462 - I-467, Washington, D.C., IEEE TAB Neural Network Committee.